# Feature-aware Label Space Dimension Reduction for Multi-label Classification

**Yao-Nan Chen**
Department of Computer Science
& Information Engineering,
National Taiwan University
r99922008@csie.ntu.edu.tw

**Hsuan-Tien Lin**
Department of Computer Science
& Information Engineering,
National Taiwan University
htlin@csie.ntu.edu.tw

## Abstract

Label space dimension reduction (LSDR) is an efficient and effective paradigm for multi-label classification with many classes. Existing approaches to LSDR, such as compressive sensing and principal label space transformation, exploit only the label part of the dataset, but not the feature part. In this paper, we propose a novel approach to LSDR that considers both the label and the feature parts. The approach, called conditional principal label space transformation, is based on minimizing an upper bound of the popular Hamming loss. The minimization step of the approach can be carried out efficiently by a simple use of singular value decomposition. In addition, the approach can be extended to a kernelized version that allows the use of sophisticated feature combinations to assist LSDR. The experimental results verify that the proposed approach is more effective than existing ones to LSDR across many real-world datasets.

## 1 Introduction

The multi-label classification problem is an extension of the traditional multiclass classification problem. In contrast to the multiclass problem, which associates only a single label to each instance, the multi-label classification problem allows multiple labels for each instance. General solutions to this problem meet the demands of many real-world applications for classifying instances into multiple concepts, including categorization of text [1], scene [2], genes [3] and so on. Given the wide range of such applications, the multi-label classification problem has been attracting much attention of researchers in machine learning [4, 5, 6].

Label space dimension reduction (LSDR) is a new paradigm in multi-label classification [4, 5]. By viewing the set of multiple labels as a high-dimensional vector in some label space, LSDR approaches use certain assumed or observed properties of the vectors to "compress" them. The compression step transforms the original multi-label classification problem (with many labels) to a small number of learning tasks. If the compression step, de-compression step, and learning steps can be efficient and effective, LSDR approaches can be useful for multi-label classification because of the appropriate use of joint information within the labels [5]. For instance, a representative LSDR approach is the principal label space transformation [PLST; 5]. PLST takes advantage of the key linear correlations between labels to build a small number of regression tasks.

LSDR approaches are homologous to the feature space dimension reduction (FSDR) approaches and share similar advantages: saving computational power and storage without much loss of prediction accuracy and improving performance by removing irrelevant, redundant, or noisy information [7]. There are two types of FSDR approaches: unsupervised and supervised. Unsupervised FSDR considers only feature information during reduction, while supervised FSDR considers the additional label information. A typical instance of unsupervised FSDR is principal component analysis [PCA; 8]. PCA transforms the features into a small number of uncorrelated variables. On the other hand, the supervised FSDR approaches include supervised principal component analysis [9], sliced inverse regression [10], and kernel dimension reduction [11]. In particular, for multi-label classification, a

leading supervised FSDR approach is canonical correlation analysis [CCA; 6, 12] which is based on linear projections in both the feature space and the label space. In general, well-tuned supervised FSDR approaches can perform better than unsupervised ones because of the additional label information.

PLST can be viewed as the counterpart of PCA in the label space [5] and is feature-unaware. That is, it considers only the label information during reduction. Motivated by the superiority of supervised FSDR over unsupervised approaches, we are interested in studying feature-aware LSDR: LSDR that considers feature information.

In this paper, we propose a novel feature-aware LSDR approach, conditional principal label space transformation (CPLST). CPLST combines the concepts of PLST (LSDR) and CCA (supervised FSDR) and can improve PLST through the addition of feature information. We derive CPLST by minimizing an upper bound of the popular Hamming loss and show that CPLST can be accomplished by a simple use of singular value decomposition. Moreover, CPLST can be flexibly extended by the kernel trick with suitable regularization, thereby allowing the use of sophisticated feature information to assist LSDR. The experimental results on real-world datasets confirm that CPLST can reduce the number of learning tasks without loss of prediction performance. In particular, CPLST is usually better than PLST and other related LSDR approaches.

The rest of this paper is organized as follows. In Section 2, we define the multi-label classification problem and review related works. Then, in Section 3, we derive the proposed CPLST approach. Finally, we present the experimental results in Section 4 and conclude our study in Section 5.

## 2 Label Space Dimension Reduction

The multi-label classification problem aims at finding a classifier from the input vector $\mathbf{x}$ to a label set $\mathcal{Y}$, where $\mathbf{x} \in \mathbb{R}^d$, $\mathcal{Y} \subseteq \{1, 2, \ldots, K\}$ and $K$ is the number of classes. The label set $\mathcal{Y}$ is often conveniently represented as a label vector, $\mathbf{y} \in \{0, 1\}^K$, where $\mathbf{y}[k] = 1$ if and only if $k \in \mathcal{Y}$. Given a dataset $\mathcal{D} = \{(\mathbf{x}_n, \mathbf{y}_n)\}_{n=1}^N$, which contains $N$ training examples $(\mathbf{x}_n, \mathbf{y}_n)$, the multi-label classification algorithm uses $\mathcal{D}$ to find a classifier $h\colon \mathcal{X} \to 2^{\{1,2,\cdots,K\}}$ anticipating that $h$ predicts $\mathbf{y}$ well on any future (unseen) test example $(\mathbf{x}, \mathbf{y})$.

There are many existing algorithms for solving multi-label classification problems. The simplest and most intuitive one is binary relevance [BR; 13]. BR decomposes the original dataset $\mathcal{D}$ into $K$ binary classification datasets, $\mathcal{D}_k = \{(\mathbf{x}_n, \mathbf{y}_n[k])\}_{n=1}^N$, and learns $K$ independent binary classifiers, each of which is learned from $\mathcal{D}_k$ and is responsible for predicting whether the label set $\mathcal{Y}$ includes label $k$. When $K$ is small, BR is an efficient and effective baseline algorithm for multi-label classification. However, when $K$ is large, the algorithm can be costly in training, prediction, and storage.

Facing the above challenges, LSDR (Label Space Dimension Reduction) offers a potential solution to these issues by compressing the $K$-dimensional label space before learning. LSDR transforms $\mathcal{D}$ into $M$ datasets, where $\mathcal{D}_m = \{(\mathbf{x}_n, \mathbf{t}_n[m])\}_{n=1}^N$, $m = 1, 2, \ldots, M$, and $M \ll K$ such that the multi-label classification problem can be tackled efficiently without significant loss of prediction performance. In particular, LSDR involves solving, predicting with, and storing the models for only $M$, instead of $K$, learning tasks.

For instance, compressive sensing [CS; 4], a precursor of LSDR, is based on the assumption that the label set vector $\mathbf{y}$ is sparse (i.e., contains few ones) to "compressed" $\mathbf{y}$ to a shorter code vector $\mathbf{t}$ by projecting $\mathbf{y}$ on $M$ random directions $\mathbf{v}_1, \cdots, \mathbf{v}_M$, where $M \ll K$ can be determined according to the assumed sparsity level. CS transforms the original multi-label classification problem into $M$ regression tasks with $\mathcal{D}_m = \{(\mathbf{x}_n, \mathbf{t}_n[m])\}_{n=1}^N$, where $\mathbf{t}_n[m] = \mathbf{v}_m^T \mathbf{y}_n$. After obtaining a multi-output regressor $\mathbf{r}(\mathbf{x})$ for predicting the code vector $\mathbf{t}$, CS decodes $\mathbf{r}(\mathbf{x})$ to the optimal label set vector by solving an optimization problem for each input instance $\mathbf{x}$ under the sparsity assumption, which can be time-consuming.

### 2.1 Principal Label Space Transformation

Principal label space transformation [PLST; 5] is another approach to LSDR. PLST first shifts each label set vector $\mathbf{y}$ to $\mathbf{z} = \mathbf{y} - \bar{\mathbf{y}}$, where $\bar{\mathbf{y}} = \frac{1}{N} \sum_{n=1}^N \mathbf{y}_n$ is the estimated mean of the label set vectors. Then, PLST takes a matrix $\mathbf{V}$ that linearly maps $\mathbf{z}$ to the code vector $\mathbf{t}$ by $\mathbf{t} = \mathbf{V}\mathbf{z}$. Unlike CS, however, PLST takes principal directions $\mathbf{v}_m$ (to be introduced next) rather than the random ones, and does not need to solve an optimization problem during decoding.

In particular, PLST considers only a matrix $\mathbf{V}$ with orthogonal rows, and decodes $\mathbf{r}(\mathbf{x})$ to the predicted labels by $\mathbf{h}(\mathbf{x}) = \text{round}(\mathbf{V}^T \mathbf{r}(\mathbf{x}) + \bar{\mathbf{y}})$, which is called round-based decoding. Tai and Lin [5] prove that when using round-based decoding and a linear transformation $\mathbf{V}$ that contains orthogonal rows, the common Hamming loss for evaluating multi-label classifiers [14] is bounded by

$$\text{Training Hamming Loss} \le c \left( \left\| \mathbf{r}(\mathbf{X}) - \mathbf{Z}\mathbf{V}^T \right\|_F^2 + \left\| \mathbf{Z} - \mathbf{Z}\mathbf{V}^T\mathbf{V} \right\|_F^2 \right), \quad (1)$$

where $\mathbf{r}(\mathbf{X})$ contains $\mathbf{r}(\mathbf{x}_n)^T$ as rows, $\mathbf{Z}$ contains $\mathbf{z}_n^T$ as rows and $c$ is a constant that depends on $K$ and $N$. The matrix $\mathbf{Z}\mathbf{V}^T$ then contains the code vector $\mathbf{t}_n^T$ as rows.

The bound can be divided into two parts. The first part is $\|\mathbf{r}(\mathbf{X}) - \mathbf{Z}\mathbf{V}^T\|_F^2$, which represents the prediction error from the regressor $\mathbf{r}(\mathbf{x}_n)$ to the desired code vectors $\mathbf{t}_n$. The second part is $\|\mathbf{Z} - \mathbf{Z}\mathbf{V}^T\mathbf{V}\|_F^2$, which stands for the encoding error for projecting $\mathbf{z}_n$ into the closest vector in $span\{\mathbf{v}_1, \cdots, \mathbf{v}_M\}$, which is $\mathbf{V}^T t_n$.

PLST is derived by minimizing the encoding error [5] and finds the optimal $M$ by $K$ matrix $\mathbf{V}$ by applying the singular value decomposition on $\mathbf{Z}$ and take the $M$ right-singular vectors $\mathbf{v}_m$ that correspond to the $M$ largest singular values. The $M$ right-singular vectors are called the principal directions for representing $\mathbf{z}_n$.

PLST can be viewed as a linear case of the kernel dependency estimation (KDE) algorithm [15]. Nevertheless, the general nonlinear KDE must solve a computationally expensive pre-image problem for each test input $\mathbf{x}$ during the prediction phase. The linearity of PLST avoids the pre-image problem and enjoys efficient round-based decoding. In this paper, we will focus on the linear case in order to design efficient algorithms for LSDR during both the training and prediction phases.

## 2.2 Canonical Correlation Analysis

A related technique that we will consider in this paper is canonical correlation analysis [CCA; 6], a well-known statistical technique for analyzing the linear relationship between two multi-dimensional variables. Traditionally, CCA is regarded as a FSDR approach in multi-label classification [12]. In this subsection, we discuss whether CCA can also be viewed as an LSDR approach. Formally, given an $N$ by $d$ matrix $\mathbf{X}$ with the $n$-th row being $\mathbf{x}_n^T$ (assumed to be zero mean) as well as an $N$ by $K$ matrix $\mathbf{Z}$ with the $n$-th row being $\mathbf{z}_n^T$ (assumed to be zero mean), CCA aims at finding two lists of basis vectors, $(\mathbf{w}_x^{(1)}, \mathbf{w}_x^{(2)}, \cdots)$ and $(\mathbf{w}_z^{(1)}, \mathbf{w}_z^{(2)}, \cdots)$, such that the correlation coefficient between the canonical variables $\mathbf{c}_x^{(i)} = \mathbf{X}\mathbf{w}_x^{(i)}$ and $\mathbf{c}_z^{(i)} = \mathbf{Z}\mathbf{w}_z^{(i)}$ is maximized, under the constraint that $\mathbf{c}_x^{(i)}$ is uncorrelated to all other $\mathbf{c}_x^{(j)}$ and $\mathbf{c}_z^{(j)}$ for $1 \le j < i$. Kettenring [16] showed that CCA is equivalent to simultaneously solving the following constrained optimization problem:

$$\min_{\mathbf{W}_x, \mathbf{W}_z} \quad \left\| \mathbf{X}\mathbf{W}_x^T - \mathbf{Z}\mathbf{W}_z^T \right\|_F^2 \qquad \text{subject to} \quad \mathbf{W}_x\mathbf{X}^T\mathbf{X}\mathbf{W}_x^T = \mathbf{W}_z\mathbf{Z}^T\mathbf{Z}\mathbf{W}_z^T = \mathbf{I}, \quad (2)$$

where $\mathbf{W}_x$ is the matrix with the $i$-th row $\left(\mathbf{w}_x^{(i)}\right)^T$, and $\mathbf{W}_z$ is the matrix with the $i$-th row $\left(\mathbf{w}_z^{(i)}\right)^T$. When CCA is considered in the context of multi-label classification, $\mathbf{X}$ is the matrix that contains mean-shifted $\mathbf{x}_n^T$ as rows and $\mathbf{Z}$ is the shifted label matrix that contains the mean-shifted $\mathbf{y}_n^T$ as rows. Traditionally, CCA is used as a supervised FSDR approach that discards $\mathbf{W}_z$ and uses only $\mathbf{W}_x$ to project features onto a lower-dimension space before learning with binary relevance [12, 17].

On the other hand, due to the symmetry between $\mathbf{X}$ and $\mathbf{Z}$, we can also view CCA as an approach to feature-aware LSDR. In particular, CCA is equivalent to first seeking projection directions $\mathbf{W}_z$ of $\mathbf{Z}$, and then performing a multi-output linear regression from $\mathbf{x}_n$ to $\mathbf{W}_z\mathbf{z}_n$, under the constraints $\mathbf{W}_x\mathbf{X}^T\mathbf{X}\mathbf{W}_x^T = \mathbf{I}$, to obtain $\mathbf{W}_x$. However, it has not been seriously studied how to use CCA for LSDR because $\mathbf{W}_z$ does not contain orthogonal rows. That is, unlike PLST, round-based decoding cannot be used and it remains to be an ongoing research issue for designing a suitable decoding scheme with CCA [18].

## 3 Proposed Algorithm

Inspired by CCA, we first design a variant that involves an appropriate decoding step. As suggested in Section 2.2, CCA is equivalent to finding a projection that minimizes the squared prediction error under the constraints $\mathbf{W}_x\mathbf{X}^T\mathbf{X}\mathbf{W}_x^T = \mathbf{W}_z\mathbf{Z}^T\mathbf{Z}\mathbf{W}_z^T = \mathbf{I}$. If we drop the constraint on $\mathbf{W}_x$ in order to further decrease the squared prediction error and change $\mathbf{W}_z\mathbf{Z}^T\mathbf{Z}\mathbf{W}_z^T = \mathbf{I}$ to $\mathbf{W}_z\mathbf{W}_z^T = \mathbf{I}$ in

order to enable round-based decoding, we obtain

$$\min_{\mathbf{W}_x, \mathbf{W}_z} \quad \left\| \mathbf{X}\mathbf{W}_x^T - \mathbf{Z}\mathbf{W}_z^T \right\|_F^2 \qquad\qquad \text{subject to} \quad \mathbf{W}_z\mathbf{W}_z^T = \mathbf{I} \tag{3}$$

Problem (3) preserves the original objective function of CCA and specifies that $\mathbf{W}_z$ must contain orthogonal rows for applying round-based decoding. We call this algorithm orthogonally constrained CCA (OCCA). Then, using the Hamming loss bound (1), when $\mathbf{V} = \mathbf{W}_z$ and $\mathbf{r}(\mathbf{x}) = \mathbf{X}\mathbf{W}_z^T$, OCCA minimizes $\|\mathbf{r}(\mathbf{x}) - \mathbf{Z}\mathbf{W}_z^T\|$ in (1) with the hope that the Hamming loss is also minimized. In other words, OCCA is employed for the orthogonal directions $\mathbf{V}$ that are "easy to learn" (of low prediction error) in terms of linear regression.

For every fixed $\mathbf{W}_z = \mathbf{V}$ in (3), the optimization problem for $\mathbf{W}_x$ is simply a linear regression from $\mathbf{X}$ to $\mathbf{Z}\mathbf{V}^T$. Then, the optimal $\mathbf{W}_x$ can be computed by a closed-form solution $\mathbf{W}_x^T = \mathbf{X}^\dagger \mathbf{Z}\mathbf{V}^T$, where $\mathbf{X}^\dagger$ is the pseudo inverse of $\mathbf{X}$. When the optimal $\mathbf{W}_x$ is inserted back into (3), the optimization problem becomes $\min_{\mathbf{V}\mathbf{V}^T=\mathbf{I}} \left\| \mathbf{X}\mathbf{X}^\dagger\mathbf{Z}\mathbf{V}^T - \mathbf{Z}\mathbf{V}^T \right\|_F^2$ which is equivalent to

$$\min_{\mathbf{V}\mathbf{V}^T=\mathbf{I}} \operatorname{tr}\left( \mathbf{V}\mathbf{Z}^T \left( \mathbf{I} - \mathbf{H} \right) \mathbf{Z}\mathbf{V}^T \right). \tag{4}$$

The matrix $\mathbf{H} = \mathbf{X}\mathbf{X}^\dagger$ is called the hat matrix for linear regression [19]. Similar to PLST, by Eckart-Young theorem [20], we can solve problem (4) by considering the eigenvectors that correspond to the largest eigenvalues of $\mathbf{Z}^T(\mathbf{H} - \mathbf{I})\mathbf{Z}$.

## 3.1 Conditional Principal Label Space Transformation

From the previous discussions, OCCA captures the input-output relation to minimize the prediction error in bound (1) with the "easy" directions. In contrast, PLST minimizes the encoding error in bound (1) with the "principal" directions. Now, we combine the benefits of the two algorithms, and minimize the two error terms simultaneously with the "conditional principal" directions. We begin by continuing our derivation of OCCA, which obtains $\mathbf{r}(\mathbf{x})$ by a linear regression from $\mathbf{X}$ to $\mathbf{Z}\mathbf{V}^T$. If we minimize both terms in (1) together with such a linear regression, the optimization problem becomes

$$\min_{\mathbf{W}, \mathbf{V}\mathbf{V}^T=\mathbf{I}} c \left( \left\| \mathbf{X}\mathbf{W}^T - \mathbf{Z}\mathbf{V}^T \right\|_F^2 + \left\| \mathbf{Z} - \mathbf{Z}\mathbf{V}^T\mathbf{V} \right\|_F^2 \right)$$

$$\Rightarrow \quad \min_{\mathbf{V}\mathbf{V}^T=\mathbf{I}} \operatorname{tr}\left( \mathbf{V}\mathbf{Z}^T \left( \mathbf{I} - \mathbf{H} \right) \mathbf{Z}\mathbf{V}^T - \mathbf{V}^T\mathbf{V}\mathbf{Z}^T\mathbf{Z} - \mathbf{Z}^T\mathbf{Z}\mathbf{V}^T\mathbf{V} + \mathbf{V}^T\mathbf{V}\mathbf{Z}^T\mathbf{Z}\mathbf{V}^T\mathbf{V} \right) \tag{5}$$

$$\Rightarrow \quad \max_{\mathbf{V}\mathbf{V}^T=\mathbf{I}} \operatorname{tr}\left( \mathbf{V}\mathbf{Z}^T\mathbf{H}\mathbf{Z}\mathbf{V}^T \right) \tag{6}$$

Problem (6) is derived by a cyclic permutation to eliminate a pair of $\mathbf{V}$ and $\mathbf{V}^T$ and combine the last three terms of (5). The problem can again be solved by taking the eigenvectors with the largest eigenvalues of $\mathbf{Z}^T\mathbf{H}\mathbf{Z}$ as the rows of $\mathbf{V}$. Such a matrix $\mathbf{V}$ minimizes the prediction error term and the encoding error term simultaneously. The resulting algorithm is called conditional principal label space transformation (CPLST), as shown in Algorithm 1.

---

**Algorithm 1** Conditional Principal Label Space Transformation

---

1: Let $\mathbf{Z} = [\mathbf{z}_1 \ldots \mathbf{z}_N]^T$ with $\mathbf{z}_n = \mathbf{y}_n - \bar{\mathbf{y}}$.
2: Preform SVD on $\mathbf{Z}^T\mathbf{H}\mathbf{Z}$ to obtain $\mathbf{Z}^T\mathbf{H}\mathbf{Z} = \mathbf{A}\Sigma\mathbf{B}$ with $\sigma_1 \geq \sigma_2 \geq \cdots \geq \sigma_N$. Let $\mathbf{V}_M$ contain the top $M$ rows of $\mathbf{B}$.
3: Encode $\{(\mathbf{x}_n, \mathbf{y}_n)\}_{n=1}^N$ to $\{(\mathbf{x}_n, \mathbf{t}_n)\}_{n=1}^N$, where $\mathbf{t_n} = \mathbf{V}_M\mathbf{z}_n$.
4: Learn a multi-dimension regressor $\mathbf{r}(\mathbf{x})$ from $\{(\mathbf{x}_n, \mathbf{t}_n)\}_{n=1}^N$.
5: Predict the label-set of an instance $\mathbf{x}$ by $\mathbf{h}(\mathbf{x}) = \operatorname{round}\left( \mathbf{V}_M^T\mathbf{r}(\mathbf{x}) + \bar{\mathbf{y}} \right)$.

---

CPLST balances the prediction error with the encoding error and is closely related with bound (1). Moreover, in contrast with PLST, which uses the key unconditional correlations, CPLST is feature-aware and allows the capture of conditional correlations [14].

We summarize the three algorithms in Table 1, and we will compare them empirically in Section 4. The three algorithms are similar. They all operate with an SVD (or eigenvalue decomposition) on a $K$ by $K$ matrix. PLST focuses on the encoding error and does not consider the features during LSDR, i.e. it is feature-unaware. On the other hand, CPLST and OCCA are feature-aware approaches, which consider features during LSDR. When using linear regression as the multi-output

Table 1: Summary of three LSDR algorithms

| Algorithm | Matrix for SVD | LSDR | Relation to bound (1) |
|-----------|----------------|------|-----------------------|
| PLST | $\mathbf{Z}^T\mathbf{Z}$ | feature-unaware | minimizes the encoding error |
| OCCA | $\mathbf{Z}^T(\mathbf{H}-\mathbf{I})\mathbf{Z}$ | feature-aware | minimizes the prediction error |
| CPLST | $\mathbf{Z}^T\mathbf{H}\mathbf{Z}$ | feature-aware | minimizes both |

regressor, CPLST simultaneously minimizes the two terms in bound (1), while OCCA minimizes only one term of the bound.

In contrast to PLST, the two feature-aware approaches OCCA and CPLST must calculate the matrix $\mathbf{H}$ and are thus slower than PLST if the dimension $d$ of the input space is large.

## 3.2  Kernelization and Regularization

Kernelization—extending a linear model to a nonlinear one using the kernel trick [21]—and regularization are two important techniques in machine learning. The former expands the power of the linear models while the latter regularizes the complexity of the learning model. In this subsection, we show that kernelization and regularization can be applied to CPLST (and OCCA).

In Section 3.1, we derive CPLST by using linear regression as the underlying multi-output regression method. Next, we replace linear regression by its kernelized form with $\ell_2$ regularization, kernel ridge regression [22], as the underlying regression algorithm. Kernel ridge regression considers a feature mapping $\Phi : \mathcal{X} \to \mathcal{F}$ before performing regularized linear regression. According to $\Phi$, the kernel function $k(\mathbf{x},\mathbf{x}') = \Phi(\mathbf{x})^T\Phi(\mathbf{x}')$ is defined as the inner product in the space $\mathcal{F}$. When applying kernel ridge regression with a regularization parameter $\lambda$ to map from $\mathbf{X}$ to $\mathbf{ZV}$, if $\Phi(\mathbf{x})$ can be explicitly computed, it is known that the closed-form solution is [22]

$$\mathbf{W} = \mathbf{\Phi}^T \left(\lambda\mathbf{I} + \mathbf{\Phi}\mathbf{\Phi}^T\right)^{-1} \mathbf{Z}\mathbf{V}^T = \mathbf{\Phi}^T \left(\lambda\mathbf{I} + \mathbf{K}\right)^{-1} \mathbf{Z}\mathbf{V}^T, \tag{7}$$

where $\mathbf{\Phi}$ is the matrix containing $\Phi(\mathbf{x}_n)^T$ as rows, and $\mathbf{K}$ is the matrix with $\mathbf{K}_{ij} = k(\mathbf{x}_i,\mathbf{x}_j) = \Phi(\mathbf{x}_i)^T\Phi(\mathbf{x}_j)$. That is, $\mathbf{K} = \mathbf{\Phi}\mathbf{\Phi}^T$ and is called the kernel matrix of $\mathbf{X}$.

Now, we derive kernel-CPLST by inserting the optimal $\mathbf{W}$ into the Hamming loss bound (1). When substituting (7) into minimizing the loss bound (1) with $r(\mathbf{X}) = \mathbf{\Phi}\mathbf{W}$ and letting $\mathbf{Q} = (\lambda\mathbf{I}+\mathbf{K})^{-1}$,

$$\min_{\mathbf{V}\mathbf{V}^T=\mathbf{I}} c \left( \left\|\mathbf{\Phi}\mathbf{\Phi}^T\mathbf{Q}\mathbf{Z}\mathbf{V}^T - \mathbf{Z}\mathbf{V}^T\right\|_F^2 + \left\|\mathbf{Z} - \mathbf{Z}\mathbf{V}^T\mathbf{V}\right\|_F^2 \right)$$

$$\Rightarrow \min_{\mathbf{V}\mathbf{V}^T=\mathbf{I}} \left( \left\|\mathbf{K}\mathbf{Q}\mathbf{Z}\mathbf{V}^T - \mathbf{Z}\mathbf{V}^T\right\|_F^2 + \left\|\mathbf{Z} - \mathbf{Z}\mathbf{V}^T\mathbf{V}\right\|_F^2 \right)$$

$$\Rightarrow \max_{\mathbf{V}\mathbf{V}^T=\mathbf{I}} \mathrm{tr}\left(\mathbf{V}\mathbf{Z}^T\left(2\mathbf{K}\mathbf{Q} - \mathbf{Q}\mathbf{K}\mathbf{K}\mathbf{Q} - \mathbf{I}\right)\mathbf{Z}\mathbf{V^T}\right) \tag{8}$$

Notice that in equation (8), kernel-CPLST do not need to explicitly compute the matrix $\mathbf{\Phi}$ and only needs the kernel matrix $\mathbf{K}$ (that can be computed through the kernel function $k$). Therefore, a high or even an infinite dimensional feature transform can be used to assist LSDR in kernel-CPLST through a suitable kernel function. Problem (8) can again be solved by considering the eigenvectors with the largest eigenvalues of $\mathbf{Z}^T\left(2\mathbf{K}\mathbf{Q} - \mathbf{Q}\mathbf{K}\mathbf{K}\mathbf{Q}\right)\mathbf{Z}$ as the rows of $\mathbf{V}$.

# 4  Experiment

In this section, we conduct experiments on eight real-world datasets, downloaded from Mulan [23], to validate the performance of CPLST and other LSDR approaches. Table 2 shows the number of labels of each dataset. Because kernel ridge regression itself, kernel-CPLST need to invert an $N$ by $N$ matrix, we can only afford to conduct a fair comparison using mid-sized datasets. In each run of the experiment, we randomly sample 80% of the dataset for training and reserve the rest for testing. All the results are reported with the mean and the standard error over 100 different random runs.

Table 2: The number of labels of each dataset

| Dataset | bib. | cor. | emo. | enr. | gen. | med. | sce. | yea. |
|---------|------|------|------|------|------|------|------|------|
| # Labels ($K$) | 159 | 374 | 6 | 53 | 27 | 45 | 6 | 14 |

We take PLST, OCCA, CPLST, and kernel-CPLST in our comparison. We do not include Compressive Sensing [13] in the comparison because earlier work [24] has shown that the algorithm is more sophisticated while being inferior to PLST. We conducted some side experiments on CCA [6] for LSDR (see Subsection 2.2) and found that it is at best comparable to OCCA. Given the space

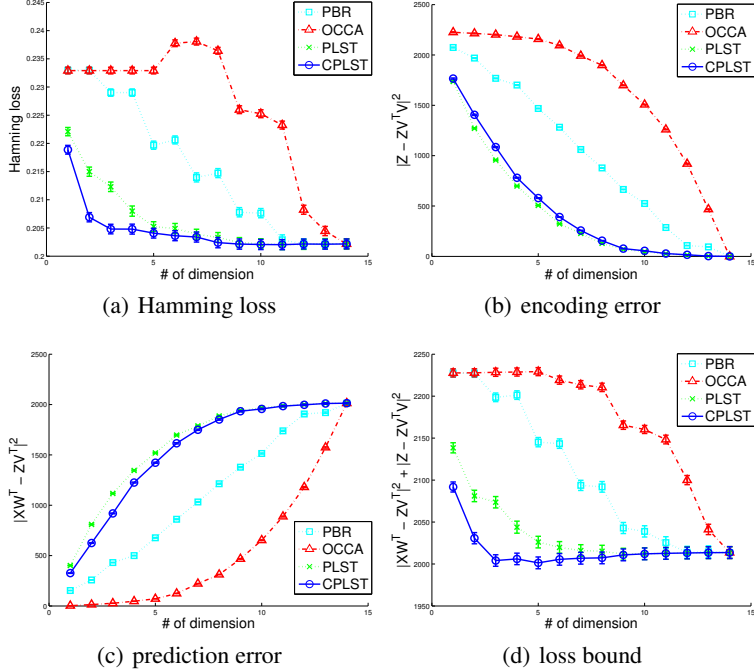

(a) Hamming loss  (b) encoding error

(c) prediction error  (d) loss bound

Figure 1: `yeast`: test results of LSDR algorithm when coupled with linear regression

constraints, we decide to only report the results on OCCA. In addition to those LSDR approaches, we also consider a simple baseline approach [24], partial binary relevance (PBR). PBR randomly selects $M$ labels from the original label set during training and only learns those $M$ binary classifiers for prediction. For the other labels, PBR directly predicts $-1$ without any training to match the sparsity assumption as exploited by Compressive Sensing [13].

### 4.1 Label Space Dimension Reduction with Linear Regression

In this subsection, we couple PBR, OCCA, PLST and CPLST with linear regression. The `yeast` dataset reveals clear differences between the four LSDR approaches and is hence taken for presentation here, while similar differences have been observed on other datasets as well. Figure 1(a) shows the test Hamming loss with respect to the possible $M$ (labels) used. It is clear that CPLST is better than the other three approaches. PLST can reach similar performance to CPLST only at a larger $M$. The other two algorithms, OCCA and PBR, are both significantly worse than CPLST.

To understand the cause of the different performance, we plot the (test) encoding error $\|\mathbf{Z} - \mathbf{Z}\mathbf{V}^T\mathbf{V}\|_F^2$, the prediction error $\|\mathbf{X}\mathbf{W}^T - \mathbf{Z}\mathbf{V}^T\|_F^2$, and the loss bound (1) in Figure 1. Figure 1(b) shows the encoding error on the test set, which matches the design of PLST. Regardless of the approaches used, the encoding error decreases to 0 when using all 14 dimensions because the $\{\mathbf{v}_m\}$'s can span the whole label space. As expected, PLST achieves the lowest encoding error across every number of dimensions. CPLST partially minimizes the encoding error in its objective function, and hence also achieves a decent encoding error. On the other hand, OCCA is blind to and hence worst at the encoding error. In particular, its encoding error is even worse than that of the baseline PBR.

Figure 1(c) shows the prediction error $\|\mathbf{X}\mathbf{W}^T - \mathbf{Z}\mathbf{V}^T\|_F^2$ on the test set, which matches the design of OCCA. First, OCCA indeed achieves the lowest prediction error across all number of dimensions. PLST, which is blind to the prediction error, reaches the highest prediction error, and is even worse than PBR. The results further reveal the trade-off between the encoding error and the prediction error: more efficient encoding of the label space are harder to predict. PLST takes the more efficient encoding to the extreme, and results in worse prediction error; OCCA, on the other hand, is better in terms of the prediction error, but leads to the least efficient encoding.

Figure 1(d) shows the scaled upper bound (1) of the Hamming loss, which equals the sum of the encoding error and the prediction error. CPLST is designed to knock down this bound, which explains its behavior in Figure 1(d) and echoes its superior performance in Figure 1(a). In fact, Figure 1(d) shows that the bound (1) is quite indicative of the performance differences in Figure 1(a). The results

Table 3: Test Hamming loss of PLST and CPLST with linear regression

| Dataset | Algorithm | $M = 20\%K$ | 40% | 60% | 80% | 100% |
|---|---|---|---|---|---|---|
| bibtex | PLST | $0.0129 \pm 0.0000$ | $0.0125 \pm 0.0000$ | $0.0124 \pm 0.0000$ | $\mathbf{0.0123 \pm 0.0000}$ | $\mathbf{0.0123 \pm 0.0000}$ |
| | CPLST | $\mathbf{0.0127 \pm 0.0000}$ | $\mathbf{0.0124 \pm 0.0000}$ | $\mathbf{0.0123 \pm 0.0000}$ | $\mathbf{0.0123 \pm 0.0000}$ | $\mathbf{0.0123 \pm 0.0000}$ |
| corel5k | PLST | $\mathbf{0.0094 \pm 0.0000}$ | $\mathbf{0.0094 \pm 0.0000}$ | $\mathbf{0.0094 \pm 0.0000}$ | $\mathbf{0.0094 \pm 0.0000}$ | $\mathbf{0.0094 \pm 0.0000}$ |
| | CPLST | $\mathbf{0.0094 \pm 0.0000}$ | $\mathbf{0.0094 \pm 0.0000}$ | $\mathbf{0.0094 \pm 0.0000}$ | $\mathbf{0.0094 \pm 0.0000}$ | $\mathbf{0.0094 \pm 0.0000}$ |
| emotions | PLST | $\mathbf{0.2207 \pm 0.0020}$ | $0.2064 \pm 0.0023$ | $0.1982 \pm 0.0022$ | $0.2013 \pm 0.0020$ | $0.2040 \pm 0.0022$ |
| | CPLST | $\mathbf{0.2189 \pm 0.0019}$ | $\mathbf{0.2059 \pm 0.0022}$ | $\mathbf{0.1990 \pm 0.0022}$ | $\mathbf{0.2015 \pm 0.0021}$ | $\mathbf{0.2040 \pm 0.0022}$ |
| enron | PLST | $\mathbf{0.0728 \pm 0.0004}$ | $0.0860 \pm 0.0005$ | $0.0946 \pm 0.0006$ | $0.1006 \pm 0.0007$ | $0.1028 \pm 0.0007$ |
| | CPLST | $\mathbf{0.0729 \pm 0.0004}$ | $0.0864 \pm 0.0005$ | $0.0943 \pm 0.0006$ | $0.1006 \pm 0.0007$ | $0.1028 \pm 0.0007$ |
| genbase | PLST | $\mathbf{0.0169 \pm 0.0004}$ | $\mathbf{0.0040 \pm 0.0002}$ | $\mathbf{0.0012 \pm 0.0001}$ | $0.0009 \pm 0.0001$ | $\mathbf{0.0007 \pm 0.0001}$ |
| | CPLST | $\mathbf{0.0168 \pm 0.0004}$ | $0.0041 \pm 0.0002$ | $\mathbf{0.0012 \pm 0.0001}$ | $\mathbf{0.0008 \pm 0.0001}$ | $\mathbf{0.0007 \pm 0.0001}$ |
| medical | PLST | $\mathbf{0.0346 \pm 0.0004}$ | $0.0407 \pm 0.0005$ | $0.0472 \pm 0.0005$ | $0.0490 \pm 0.0005$ | $0.0497 \pm 0.0006$ |
| | CPLST | $\mathbf{0.0346 \pm 0.0004}$ | $\mathbf{0.0406 \pm 0.0005}$ | $\mathbf{0.0471 \pm 0.0005}$ | $0.0490 \pm 0.0005$ | $0.0497 \pm 0.0006$ |
| scene | PLST | $0.1809 \pm 0.0004$ | $0.1718 \pm 0.0004$ | $0.1566 \pm 0.0007$ | $0.1321 \pm 0.0008$ | $\mathbf{0.1106 \pm 0.0008}$ |
| | CPLST | $\mathbf{0.1744 \pm 0.0004}$ | $\mathbf{0.1532 \pm 0.0005}$ | $\mathbf{0.1349 \pm 0.0005}$ | $\mathbf{0.1209 \pm 0.0007}$ | $\mathbf{0.1106 \pm 0.0008}$ |
| yeast | PLST | $0.2150 \pm 0.0008$ | $0.2052 \pm 0.0009$ | $0.2033 \pm 0.0009$ | $\mathbf{0.2020 \pm 0.0009}$ | $\mathbf{0.2022 \pm 0.0009}$ |
| | CPLST | $\mathbf{0.2069 \pm 0.0008}$ | $\mathbf{0.2041 \pm 0.0009}$ | $\mathbf{0.2024 \pm 0.0009}$ | $\mathbf{0.2020 \pm 0.0009}$ | $\mathbf{0.2022 \pm 0.0009}$ |

(those within one standard error of the lower one are in bold)

Table 4: Test Hamming loss of LSDR algorithm with M5P

| Dataset | Algorithm | $M = 20\%K$ | 40% | 60% | 80% | 100% |
|---|---|---|---|---|---|---|
| bibtex | PLST | $0.0130 \pm 0.0001$ | $\mathbf{0.0128 \pm 0.0001^*}$ | $\mathbf{0.0128 \pm 0.0001}$ | $\mathbf{0.0127 \pm 0.0001^*}$ | $\mathbf{0.0127 \pm 0.0001^*}$ |
| | CPLST | $\mathbf{0.0129 \pm 0.0001^*}$ | $\mathbf{0.0128 \pm 0.0001^*}$ | $\mathbf{0.0127 \pm 0.0001^*}$ | $\mathbf{0.0127 \pm 0.0001^*}$ | $\mathbf{0.0127 \pm 0.0001^*}$ |
| corel5k | PLST | $\mathbf{0.0094 \pm 0.0000^*}$ | $\mathbf{0.0094 \pm 0.0000^*}$ | $\mathbf{0.0094 \pm 0.0000^*}$ | $\mathbf{0.0094 \pm 0.0000^*}$ | $\mathbf{0.0094 \pm 0.0000^*}$ |
| | CPLST | $\mathbf{0.0094 \pm 0.0000^*}$ | $\mathbf{0.0094 \pm 0.0000^*}$ | $\mathbf{0.0094 \pm 0.0000^*}$ | $\mathbf{0.0094 \pm 0.0000^*}$ | $\mathbf{0.0094 \pm 0.0000^*}$ |
| emotions | PLST | $\mathbf{0.2213 \pm 0.0030}$ | $\mathbf{0.2109 \pm 0.0030}$ | $0.2039 \pm 0.0029$ | $0.2051 \pm 0.0029$ | $\mathbf{0.2063 \pm 0.0030}$ |
| | CPLST | $\mathbf{0.2209 \pm 0.0031^*}$ | $\mathbf{0.2085 \pm 0.0032^*}$ | $\mathbf{0.2004 \pm 0.0031^*}$ | $\mathbf{0.2020 \pm 0.0031^*}$ | $\mathbf{0.2046 \pm 0.0031^*}$ |
| enron | PLST | $\mathbf{0.0490 \pm 0.0002}$ | $\mathbf{0.0488 \pm 0.0002^*}$ | $\mathbf{0.0489 \pm 0.0002^*}$ | $\mathbf{0.0490 \pm 0.0002^*}$ | $\mathbf{0.0490 \pm 0.0002^*}$ |
| | CPLST | $\mathbf{0.0489 \pm 0.0003^*}$ | $\mathbf{0.0489 \pm 0.0003}$ | $\mathbf{0.0490 \pm 0.0003}$ | $\mathbf{0.0490 \pm 0.0003^*}$ | $\mathbf{0.0490 \pm 0.0003^*}$ |
| genbase | PLST | $\mathbf{0.0215 \pm 0.0004^*}$ | $\mathbf{0.0202 \pm 0.0004^*}$ | $\mathbf{0.0195 \pm 0.0003^*}$ | $\mathbf{0.0194 \pm 0.0003^*}$ | $\mathbf{0.0194 \pm 0.0003^*}$ |
| | CPLST | $\mathbf{0.0215 \pm 0.0004^*}$ | $\mathbf{0.0202 \pm 0.0004^*}$ | $\mathbf{0.0195 \pm 0.0003^*}$ | $0.0195 \pm 0.0003$ | $0.0195 \pm 0.0003$ |
| medical | PLST | $0.0127 \pm 0.0002$ | $\mathbf{0.0099 \pm 0.0002^*}$ | $0.0097 \pm 0.0002$ | $0.0097 \pm 0.0002$ | $0.0097 \pm 0.0002$ |
| | CPLST | $\mathbf{0.0126 \pm 0.0002^*}$ | $\mathbf{0.0099 \pm 0.0002^*}$ | $\mathbf{0.0096 \pm 0.0002^*}$ | $\mathbf{0.0096 \pm 0.0002^*}$ | $\mathbf{0.0096 \pm 0.0002^*}$ |
| scene | PLST | $0.1802 \pm 0.0005$ | $0.1688 \pm 0.0007$ | $0.1540 \pm 0.0008$ | $0.1396 \pm 0.0011$ | $0.1281 \pm 0.0008$ |
| | CPLST | $\mathbf{0.1674 \pm 0.0005}$ | $\mathbf{0.1538 \pm 0.0006^*}$ | $\mathbf{0.1428 \pm 0.0007^*}$ | $\mathbf{0.1289 \pm 0.0007^*}$ | $\mathbf{0.1268 \pm 0.0008^*}$ |
| yeast | PLST | $0.2162 \pm 0.0008$ | $0.2082 \pm 0.0009$ | $0.2071 \pm 0.0009$ | $\mathbf{0.2064 \pm 0.0009^*}$ | $0.2067 \pm 0.0009$ |
| | CPLST | $\mathbf{0.2083 \pm 0.0009^*}$ | $\mathbf{0.2064 \pm 0.0009^*}$ | $\mathbf{0.2063 \pm 0.0009^*}$ | $\mathbf{0.2064 \pm 0.0009^*}$ | $\mathbf{0.2066 \pm 0.0009^*}$ |

(those with the lowest mean are marked with *; those within one standard error of the lowest one are in bold)

demonstrate that CPLST explores the trade-off between the encoding error and the prediction error in an optimal manner to reach the best performance for label space dimension reduction.

The results of PBR and OCCA are consistently inferior to PLST and CPLST across most of the datasets in our experiments [25] and are not reported here because of space constraints. The test Hamming loss achieved by PLST and CPLST on other datasets with different percentage of used labels are reported in Table 3. In most datasets, CPLST is at least as effective as PLST; in bibtex, scene and yeast, CPLST performs significantly better than PLST.

Note that in the medical and enron datasets, both PLST and CPLST overfit when using many dimensions. That is, the performance of both algorithms would be better when using fewer dimensions (than the full binary relevance, which is provably equivalent to either PLST or CPLST with $M = K$ when using linear regression). These results demonstrate that LSDR approaches, like their feature space dimension reduction counterparts, can potentially help resolve the issue of overfitting.

## 4.2 Coupling Label Space Dimension Reduction with the M5P Decision Tree

CPLST is designed by assuming a specific regression method. Next, we demonstrate that the input-output relationship captured by CPLST is not restricted for coupling with linear regression, but can be effective for other regression methods in the learning stage (step 4 of Algorithm 1). We do so by coupling the LSDR approaches with the M5P decision tree [26]. M5P decision tree is a non-linear regression method. We take the implementation from WEKA [27] for M5P with the default parameter setting.

The experimental results are shown in Table 4. The relations between PLST and CPLST when coupled with M5P are similar to the ones when coupled with linear regression. In particular, in the yeast, scene, and emotions, CPLST outperforms PLST. The results demonstrate that the captured input-output relation is also effective for regression methods other than linear regression.

## 4.3 Label Space Dimension Reduction with Kernel Ridge Regression

In this subsection, we conduct experiments for demonstrating the performance of kernelization and regularization. For kernel-CPLST, we use the Gaussian kernel $k(\mathbf{x}_i, \mathbf{x}_j) = \exp\left(-\gamma \|\mathbf{x}_i - \mathbf{x}_j\|^2\right)$

Table 5: Test Hamming loss of LSDR algorithm with kernel ridge regression

| Dataset | Algorithm | $M = 20\%K$ | 40% | 60% | 80% | 100% |
|---|---|---|---|---|---|---|
| bibtex | PLST | $0.0151 \pm 0.0000$ | $0.0151 \pm 0.0000$ | $0.0151 \pm 0.0000$ | $0.0151 \pm 0.0000$ | $0.0151 \pm 0.0000$ |
| | kernel-CPLST | $\mathbf{0.0127 \pm 0.0000}$ | $\mathbf{0.0123 \pm 0.0000}$ | $\mathbf{0.0121 \pm 0.0000}$ | $\mathbf{0.0120 \pm 0.0000}$ | $\mathbf{0.0120 \pm 0.0000}$ |
| corel5k | PLST | $0.0094 \pm 0.0000$ | $0.0094 \pm 0.0000$ | $0.0094 \pm 0.0000$ | $0.0094 \pm 0.0000$ | $0.0094 \pm 0.0000$ |
| | kernel-CPLST | $\mathbf{0.0092 \pm 0.0000}$ | $\mathbf{0.0092 \pm 0.0000}$ | $\mathbf{0.0092 \pm 0.0000}$ | $\mathbf{0.0092 \pm 0.0000}$ | $\mathbf{0.0092 \pm 0.0000}$ |
| emotions | PLST | $\mathbf{0.2218 \pm 0.0020}$ | $0.2074 \pm 0.0023$ | $\mathbf{0.1983 \pm 0.0026}$ | $0.2000 \pm 0.0025$ | $\mathbf{0.2002 \pm 0.0025}$ |
| | kernel-CPLST | $\mathbf{0.2231 \pm 0.0020}$ | $\mathbf{0.2071 \pm 0.0024}$ | $\mathbf{0.1981 \pm 0.0025}$ | $\mathbf{0.1973 \pm 0.0027}$ | $\mathbf{0.1988 \pm 0.0027}$ |
| enron | PLST | $0.0460 \pm 0.0002$ | $0.0462 \pm 0.0002$ | $0.0466 \pm 0.0002$ | $0.0468 \pm 0.0002$ | $0.0469 \pm 0.0002$ |
| | kernel-CPLST | $\mathbf{0.0453 \pm 0.0002}$ | $\mathbf{0.0454 \pm 0.0002}$ | $\mathbf{0.0455 \pm 0.0002}$ | $\mathbf{0.0455 \pm 0.0002}$ | $\mathbf{0.0456 \pm 0.0002}$ |
| genbase | PLST | $\mathbf{0.0169 \pm 0.0004}$ | $\mathbf{0.0039 \pm 0.0002}$ | $\mathbf{0.0014 \pm 0.0001}$ | $\mathbf{0.0010 \pm 0.0001}$ | $\mathbf{0.0008 \pm 0.0001}$ |
| | kernel-CPLST | $\mathbf{0.0170 \pm 0.0004}$ | $\mathbf{0.0040 \pm 0.0002}$ | $\mathbf{0.0013 \pm 0.0001}$ | $\mathbf{0.0009 \pm 0.0001}$ | $\mathbf{0.0008 \pm 0.0001}$ |
| medical | PLST | $0.0136 \pm 0.0002$ | $0.0106 \pm 0.0002$ | $0.0103 \pm 0.0002$ | $0.0102 \pm 0.0002$ | $0.0102 \pm 0.0002$ |
| | kernel-CPLST | $\mathbf{0.0131 \pm 0.0002}$ | $\mathbf{0.0098 \pm 0.0002}$ | $\mathbf{0.0096 \pm 0.0002}$ | $\mathbf{0.0096 \pm 0.0002}$ | $\mathbf{0.0096 \pm 0.0002}$ |
| scene | PLST | $\mathbf{0.1713 \pm 0.0004}$ | $\mathbf{0.1468 \pm 0.0006}$ | $\mathbf{0.1173 \pm 0.0008}$ | $0.0932 \pm 0.0011$ | $0.0731 \pm 0.0007$ |
| | kernel-CPLST | $0.1733 \pm 0.0004$ | $\mathbf{0.1470 \pm 0.0006}$ | $\mathbf{0.1179 \pm 0.0007}$ | $\mathbf{0.0905 \pm 0.0007}$ | $\mathbf{0.0717 \pm 0.0007}$ |
| yeast | PLST | $0.2030 \pm 0.0008$ | $0.1913 \pm 0.0009$ | $0.1892 \pm 0.0009$ | $0.1882 \pm 0.0009$ | $0.1881 \pm 0.0009$ |
| | kernel-CPLST | $\mathbf{0.2018 \pm 0.0008}$ | $\mathbf{0.1904 \pm 0.0009}$ | $\mathbf{0.1875 \pm 0.0009}$ | $\mathbf{0.1869 \pm 0.0009}$ | $\mathbf{0.1868 \pm 0.0009}$ |

(those within one standard error of the lower one are in bold)

during LSDR and take kernel ridge regression with the same kernel and the same regularization parameter as the underlying multi-output regression method. We also couple PLST with kernel ridge regression for a fair comparison. We select the Gaussian kernel parameter $\gamma$ and the regularization parameter $\lambda$ with a grid search on $(\log_2 \lambda, \log_2 \gamma)$ using a 5-fold cross validation using the sum of the Hamming loss across all dimensions. The details of the grid search can be found in the Master's Thesis of the first author [25].

When coupled with kernel ridge regression, the comparison between PLST and kernel-CPLST in terms of the Hamming loss is shown in Table 5. kernel-CPLST performs well for LSDR and outperforms the feature-unaware PLST in most cases. In particular, in five out of the eight datasets, kernel-CPLST is significantly better than PLST regardless of the number of dimensions used. In addition, in the medical and enron datasets, the overfitting problem is eliminated with regularization (and parameter selection), and hence kernel-CPLST not only performs better than PLST with kernel ridge regression, but also is better than the (unregularized) linear regression results in Table 3.

From the previous comparison between CPLST and PLST, CPLST is at least as good as, and usually better than, PLST. The difference between CPLST and PLST is small but consistent, and does suggest that CPLST is a better choice for label-space dimension reduction. The results provide practical insights on the two types of label correlation [14]: unconditional correlation (feature-unaware) and conditional correlation (feature-aware). The unconditional correlation, exploited by PLST and other LSDR algorithms, readily leads to promising performance in practice. On the other hand, there is room for some (albeit small) improvements when exploiting the conditional correlation properly like CPLST.

## 5 Conclusion

In this paper, we studied feature-aware label space dimension reduction (LSDR) approaches, which utilize the feature information during LSDR and can be viewed as the counterpart of supervised feature space dimension reduction. We proposed a novel feature-aware LSDR algorithm, conditional principal label space transformation (CPLST) which utilizes the key conditional correlations for dimension reduction. CPLST enjoys the theoretical guarantee in balancing between the prediction error and the encoding error in minimizing the Hamming loss bound. In addition, we extended CPLST to a kernelized version for capturing more sophisticated relations between features and labels. We conducted experiments for comparing CPLST and its kernelized version with other LSDR approaches. The experimental results demonstrated that CPLST is the best among the LSDR approaches when coupled with linear regression or kernel ridge regression. In particular, CPLST is consistently better than its feature-unaware precursor, PLST. Moreover, the input-output relation captured by CPLST can be utilized by regression method other than linear regression.

## Acknowledgments

We thank the anonymous reviewers of the conference and members of the Computational Learning Laboratory at National Taiwan University for valuable suggestions. This work is partially supported by National Science Council of Taiwan via the grant NSC 101-2628-E-002-029-MY2.

# References

[1] I. Katakis, G. Tsoumakas, and I. Vlahavas. Multilabel text classification for automated tag suggestion. In *Proceedings of the European Conference on Machine Learning and Principles and Practice of Knowledge Discovery in Databases 2008 Discovery Challenge*, 2008.

[2] M. Boutell, J. Luo, X. Shen, and C. Brown. Learning multi-label scene classification. *Pattern Recognition*, 2004.

[3] A. Elisseeff and J. Weston. A kernel method for multi-labelled classification. In *Advances in Neural Information Processing Systems 14*, 2001.

[4] D. Hsu, S. Kakade, J. Langford, and T. Zhang. Multi-Label prediction via compressed sensing. In *Advances in Neural Information Processing Systems 22*, 2009.

[5] F. Tai and H.-T. Lin. Multi-Label classification with principal label space transformation. In *Neural Computation*, 2012.

[6] H. Hotelling. Relations between two sets of variates. *Biometrika*, 1936.

[7] M. Wall, A. Rechtsteiner, and L. Rocha. Singular value decomposition and principal component analysis. *A Practical Approach to Microarray Data Analysis*, 2003.

[8] I. Jolliffe. *Principal Component Analysis*. Springer, second edition, October 2002.

[9] E. Barshan, A. Ghodsi, Z. Azimifar, and M. Zolghadri Jahromi. Supervised principal component analysis: Visualization, classification and regression on subspaces and submanifolds. *Pattern Recognition*, 2011.

[10] K.-C. Li. Sliced inverse regression for dimension reduction. *Journal of the American Statistical Association*, 1991.

[11] K. Fukumizu, F. Bach, and M. Jordan. Dimensionality reduction for supervised learning with reproducing kernel hilbert spaces. *Journal of Machine Learning Research*, 2004.

[12] L. Sun, S. Ji, and J. Ye. Canonical correlation analysis for multilabel classification: A least-squares formulation, extensions, and analysis. *IEEE Transactions on Pattern Analysis and Machine Intelligence*, 2011.

[13] G. Tsoumakas, I. Katakis, and I. Vlahavas. Mining multi-label data. In *Data Mining and Knowledge Discovery Handbook*. Springer US, 2010.

[14] K. Dembczynski, W. Waegeman, W. Cheng, and E. Hüllermeier. On label dependence and loss minimization in multi-label classification. *Machine Learning*, 2012.

[15] J. Weston, O. Chapelle, A. Elisseeff, B. Schölkopf, and V. Vapnik. Kernel dependency estimation. In *Advances in Neural Information Processing Systems 15*, 2002.

[16] J. Kettenring. Canonical analysis of several sets of variables. *Biometrika*, 1971.

[17] S. Yu, K. Yu, V. Tresp, and H.-P. Kriegel. Multi-output regularized feature projection. *IEEE Transactions on Knowledge and Data Engineering*, 2006.

[18] Y. Zhang and J. Schneider. Multi-label output codes using canonical correlation analysis. In *Proceedings of the Fourteenth International Conference on Artificial Intelligence and Statistics*, 2011.

[19] D. Hoaglin and R. Welsch. The hat matrix in regression and ANOVA. *The American Statistician*, 1978.

[20] C. Eckart and G. Young. The approximation of one matrix by another of lower rank. *Psychometrika*, 1936.

[21] B. Schölkopf and A. Smola. *Learning with kernels : support vector machines, regularization, optimization, and beyond*. The MIT Press, first edition, 2002.

[22] G. Saunders, A. Gammerman, and V. Vovk. Ridge regression learning algorithm in dual variables. In *Proceedings of the Fifteenth International Conference on Machine Learning*, 1998.

[23] G. Tsoumakas, E. Spyromitros-Xioufis, J. Vilcek, and I. Vlahavas. Mulan: A java library for multi-label learning. *Journal of Machine Learning Research*, 2011.

[24] B. Datta. *Numerical Linear Algebra and Applications, Second Edition*. SIAM-Society for Industrial and Applied Mathematics, 2010.

[25] Y.-N. Chen. Feature-aware label space dimension reduction for multi-label classification problem. Master's thesis, National Taiwan University, 2012.

[26] Y. Wang and I. Witten. Induction of model trees for predicting continuous classes. In *Poster Papers of the Nineth European Conference on Machine Learning*, 1997.

[27] M. Hall, E. Frank, G. Holmes, B. Pfahringer, P. Reutemann, and I. Witten. The weka data mining software: an update. *SIGKDD Exploration Newsletter*, 2009.

